# A Framework for Multiple-Instance Learning

**Oded Maron**
NE43-755
AI Lab, M.I.T.
Cambridge, MA 02139
oded@ai.mit.edu

**Tomás Lozano-Pérez**
NE43-836a
AI Lab, M.I.T.
Cambridge, MA 02139
tlp@ai.mit.edu

## Abstract

Multiple-instance learning is a variation on supervised learning, where the task is to learn a concept given positive and negative *bags* of instances. Each bag may contain many instances, but a bag is labeled positive even if only one of the instances in it falls within the concept. A bag is labeled negative only if all the instances in it are negative. We describe a new general framework, called *Diverse Density*, for solving multiple-instance learning problems. We apply this framework to learn a simple description of a person from a series of images (bags) containing that person, to a stock selection problem, and to the drug activity prediction problem.

## 1 Introduction

One of the drawbacks of applying the supervised learning model is that it is not always possible for a teacher to provide labeled examples for training. Multiple-instance learning provides a new way of modeling the teacher's weakness. Instead of receiving a set of instances which are labeled positive or negative, the learner receives a set of *bags* that are labeled positive or negative. Each bag contains many instances. A bag is labeled negative if all the instances in it are negative. On the other hand, a bag is labeled positive if there is at least one instance in it which is positive. From a collection of labeled bags, the learner tries to induce a concept that will label individual instances correctly. This problem is harder than even noisy supervised learning since the ratio of negative to positive instances in a positively-labeled bag (the noise ratio) can be arbitrarily high.

The first application of multiple-instance learning was to drug activity prediction. In the activity prediction application, one objective is to predict whether a candidate drug molecule will bind strongly to a target protein known to be involved in some disease state. Typically,

one has examples of molecules that bind well to the target protein and also of molecules that do not bind well. Much as in a lock and key, shape is the most important factor in determining whether a drug molecule and the target protein will bind. However, drug molecules are flexible, so they can adopt a wide range of shapes. A positive example does not convey what shape the molecule took in order to bind – only that *one* of the shapes that the molecule can take was the right one. However, a negative example means that none of the shapes that the molecule can achieve was the right key.

The multiple-instance learning model was only recently formalized by [Dietterich *et al.*, 1997]. They assume a hypothesis class of axis-parallel rectangles, and develop algorithms for dealing with the drug activity prediction problem described above. This work was followed by [Long and Tan, 1996], where a high-degree polynomial PAC bound was given for the number of examples needed to learn in the multiple-instance learning model. [Auer, 1997] gives a more efficient algorithm, and [Blum and Kalai, 1998] shows that learning from multiple-instance examples is reducible to PAC-learning with two sided noise and to the Statistical Query model. Unfortunately, the last three papers make the restrictive assumption that all instances from all bags are generated independently.

In this paper, we describe a framework called *Diverse Density* for solving multiple-instance problems. Diverse Density is a measure of the intersection of the positive bags minus the union of the negative bags. By maximizing Diverse Density we can find the point of intersection (the desired concept), and also the set of feature weights that lead to the best intersection. We show results of applying this algorithm to a difficult synthetic training set as well as the "musk" data set from [Dietterich *et al.*, 1997]. We then use Diverse Density in two novel applications: one is to learn a simple description of a person from a series of images that are labeled positive if the person is somewhere in the image and negative otherwise. The other is to deal with a high amount of noise in a stock selection problem.

## 2  Diverse Density

We motivate the idea of Diverse Density through a molecular example. Suppose that the shape of a candidate molecule can be adequately described by a feature vector . One instance of the molecule is therefore represented as a point in n-dimensional feature space. As the molecule changes its shape (through both rigid and non-rigid transformations), it will trace out a manifold through this n-dimensional space[1]. Figure 1(a) shows the paths of four molecules through a 2-dimensional feature space.

If a candidate molecule is labeled positive, we know that in at least one place along the manifold, it took on the right shape for it to fit into the target protein. If the molecule is labeled negative, we know that none of the conformations along its manifold will allow binding with the target protein. If we assume that there is only one shape that will bind to the target protein, what do the positive and negative manifolds tell us about the location of the correct shape in feature space? The answer: it is where all positive feature-manifolds intersect without intersecting any negative feature-manifolds. For example, in Figure 1(a) it is point A.

Unfortunately, a multiple-instance bag does not give us complete distribution information, but only some arbitrary sample from that distribution. In fact, in applications other than drug discovery, there is not even a notion of an underlying continuous manifold. Therefore, Figure 1(a) becomes Figure 1(b). The problem of trying to find an intersection changes

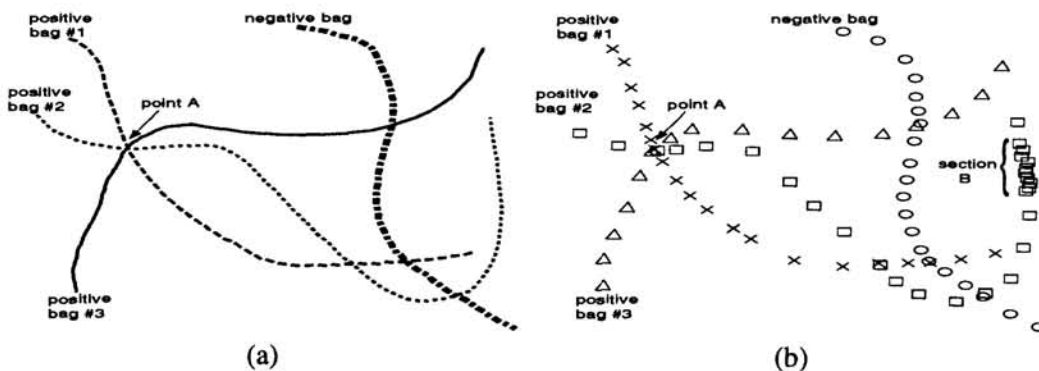

|                        |                        |
| ---------------------- | ---------------------- |
| (a)                    | (b)                    |

The different shapes that a molecule can take on are represented as a path. The intersection point of positive paths is where they took on the same shape.

Samples taken along the paths. Section B is a high density area, but point A is a high Diverse Density area.

Figure 1: A motivating example for Diverse Density

to a problem of trying to find an area where there is both high density of positive points and low density of negative points. The difficulty with using regular density is illustrated in Figure 1(b), Section B. We are not just looking for high density, but high "Diverse Density". We define Diverse Density at a point to be a measure of how many *different* positive bags have instances near that point, and how far the negative instances are from that point.

### 2.1 Algorithms for multiple-instance learning

In this section, we derive a probabilistic measure of Diverse Density, and test it on a difficult artificial data set. We denote positive bags as $B_i^+$, the $j^{th}$ point in that bag as $B_{ij}^+$, and the value of the $k^{th}$ feature of that point as $B_{ijk}^+$. Likewise, $B_{ij}^-$ represents a negative point. Assuming for now that the true concept is a single point $t$, we can find it by maximizing $\Pr(x = t \mid B_1^+, \cdots, B_n^+, B_1^-, \cdots, B_m^-)$ over all points $x$ in feature space. If we use Bayes' rule and an uninformative prior over the concept location, this is equivalent to maximizing the likelihood $\Pr(B_1^+, \cdots, B_n^+, B_1^-, \cdots, B_m^- \mid x = t)$. By making the additional assumption that the bags are conditionally independent given the target concept $t$, the best hypothesis is $\arg\max_x \prod_i \Pr(B_i^+ \mid x = t) \prod_i \Pr(B_i^- \mid x = t)$. Using Bayes' rule once more (and again assuming a uniform prior over concept location), this is equivalent to

$$\arg \max_x \prod_i \Pr(x = t \mid B_i^+) \prod_i \Pr(x = t \mid B_i^-). \tag{1}$$

This is a general definition of maximum Diverse Density, but we need to define the terms in the products to instantiate it. One possibility is a noisy-or model: the probability that not all points missed the target is $\Pr(x = t \mid B_i^+) = \Pr(x = t \mid B_{i1}^+, B_{i2}^+, \ldots) = 1 - \prod_j (1 - \Pr(x = t \mid B_{ij}^+))$, and likewise $\Pr(x = t \mid B_i^-) = \prod_j (1 - \Pr(x = t \mid B_{ij}^-))$. We model the causal probability of an individual instance on a potential target as related to the distance between them. Namely, $\Pr(x = t \mid B_{ij}) = \exp(- \parallel B_{ij} - x \parallel^2)$. Intuitively, if one of the instances in a positive bag is close to $x$, then $\Pr(x = t \mid B_i^+)$ is high. Likewise, if every positive bag has an instance close to $x$ and no negative bags are close to $x$, then $x$ will have high Diverse Density. Diverse Density at an intersection of $n$ bags is exponentially higher than it is at an intersection of $n - 1$ bags, yet all it takes is one well placed negative instance to drive the Diverse Density down.

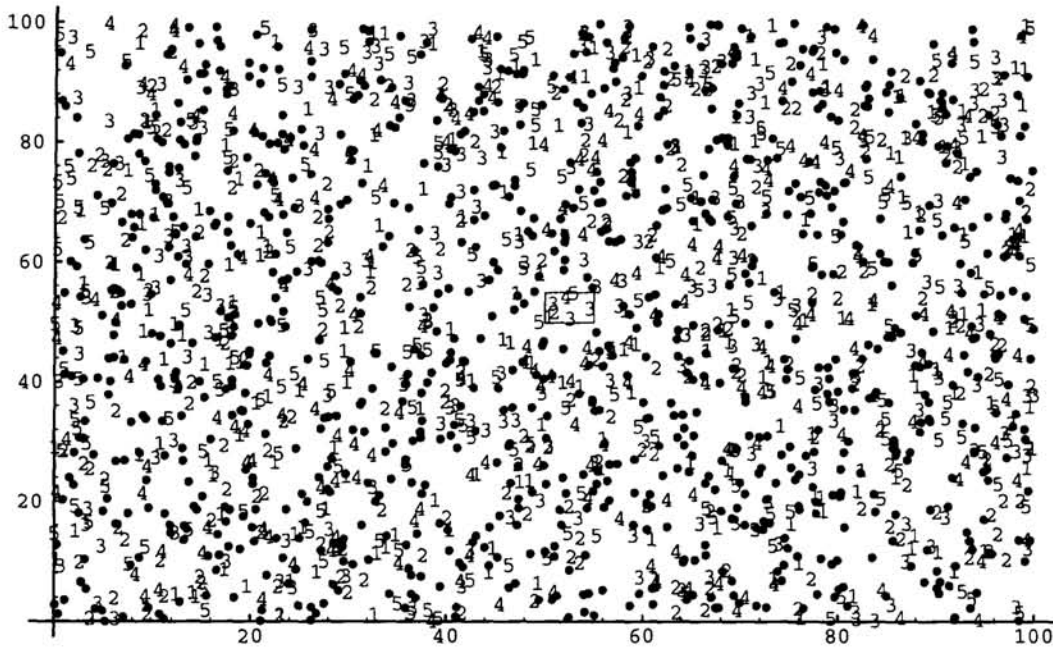

Figure 2: Negative and positive bags drawn from the same distribution, but labeled according to their intersection with the middle square. Negative instances are dots, positive are numbers. The square contains at least one instance from every positive bag and no negatives,

The Euclidean distance metric used to measure "closeness" depends on the features that describe the instances. It is likely that some of the features are irrelevant, or that some should be weighted to be more important than others. Luckily, we can use the same framework to find not only the best location in feature space, but also the best weighting of the features. Once again, we find the best scaling of the individual features by finding the scalings that maximize Diverse Density. The algorithm returns both a location $x$ and a scaling vector $s$, where $\| B_{ij} - x \|^2 = \sum_k s_k^2 (B_{ijk} - x_k)^2$.

Note that the assumption that all bags intersect at a single point is not necessary. We can assume more complicated concepts, such as for example a disjunctive concept $t_a \vee t_b$. In this case, we maximize over a pair of locations $x_a$ and $x_b$ and define $\Pr(x_a = t_a \vee x_b = t_b \mid B_{ij}) = \max_{x_a, x_b}(\Pr(x_a = t_a \mid B_{ij}), \Pr(x_b = t_b \mid B_{ij}))$.

To test the algorithm, we created an artificial data set: 5 positive and 5 negative bags, each with 50 instances. Each instance was chosen uniformly at randomly from a $[0, 100] \times [0, 100] \in \mathcal{R}^2$ domain, and the concept was a $5 \times 5$ square in the middle of the domain. A bag was labeled positive if at least one of its instances fell within the square, and negative if none did, as shown in Figure 2. The square in the middle contains at least one instance from every positive bag and no negative instances. This is a difficult data set because both positive and negative bags are drawn from the same distribution. They only differ in a small area of the domain.

Using regular density (adding up the contribution of every positive bag and subtracting negative bags; this is roughly what a supervised learning algorithm such as nearest neighbor performs), we can plot the density surface across the domain. Figure 3(a) shows this surface for the data set in Figure 2, and it is clear that finding the peak (a candidate hypothesis) is difficult. However, when we plot the Diverse Density surface (using the noisy-or model) in Figure 3(b), it is easy to pick out the global maximum which is within the desired concept. The other

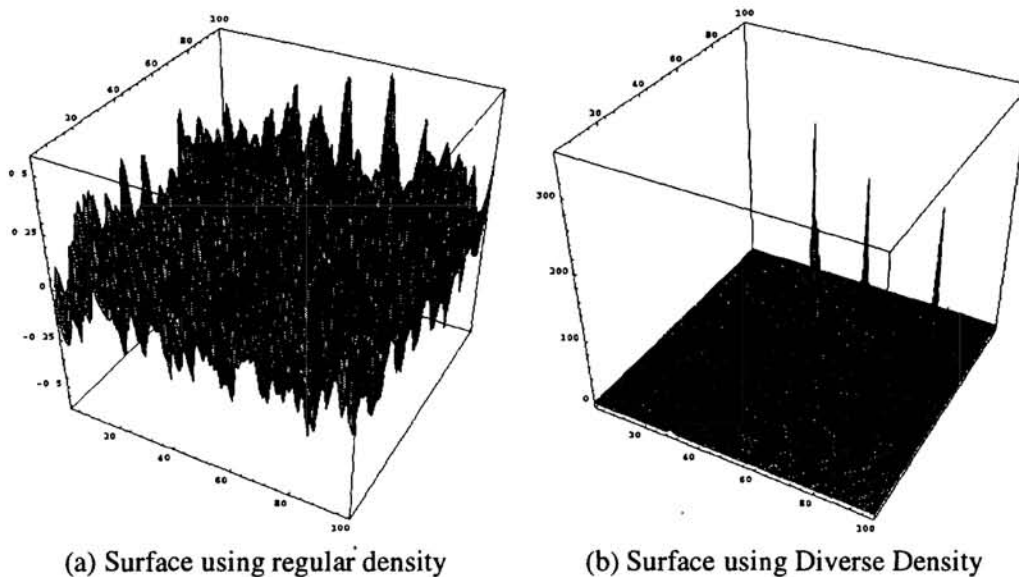

(a) Surface using regular density          (b) Surface using Diverse Density

Figure 3: Density surfaces over the example data of Figure 3

major peaks in Figure 3(b) are the result of a chance concentration of instances from different bags. With a bit more bad luck, one of those peaks could have eclipsed the one in the middle. However, the chance of this decreases as the number of bags (training examples) increases.

One remaining issue is how to find the maximum Diverse Density. In general, we are searching an arbitrary density landscape and the number of local maxima and size of the search space could prohibit any efficient exploration. In this paper, we use gradient ascent with multiple starting points. This has worked succesfully in every test case because we know what starting points to use. The maximum Diverse Density peak is made of contributions from some set of positive points. If we start an ascent from every positive point, one of them is likely to be closest to the maximum, contribute the most to it and have a climb directly to it. While this heuristic is sensible for maximizing with respect to location, maximizing with respect to scaling of feature weights may still lead to local maxima.

## 3  Applications of Diverse Density

By way of benchmarking, we tested the Diverse Density approach on the "musk" data sets from [Dietterich *et al.*, 1997], which were also used in [Auer, 1997]. We also have begun investigating two new applications of multiple-instance learning. We describe preliminary results on all of these below. The musk data sets contain feature vectors describing the surfaces of a variety of low-energy shapes from approximately 100 molecules. Each feature vector has 166 dimensions. Approximately half of these molecules are known to smell "musky," the remainder are very similar molecules that do not smell musky. There are two musk data sets; the Musk-1 data set is smaller, both in having fewer molecules and many fewer instances per molecule. Many (72) of the molecules are shared between the two data sets, but the second set includes more instances for the shared molecules.

We approached the problem as follows: for each run, we held out a randomly selected 1/10 of the data set as a test set. We computed the maximum Diverse Density on the training set by multiple gradient ascents, starting at each positive instance. This produces a

maximum feature point as well as the best feature weights corresponding to that point. We note that typically less than half of the 166 features receive non-zero weighting. We then computed a distance threshold that optimized classification performance under leave-one-out cross validation within the training set. We used the feature weights and distance threshold to classify the examples of the test set; an example was deemed positive if the weighted distance from the maximum density point to any of its instances was below the threshold.

The table below lists the average accuracy of twenty runs, compared with the performance of the two principal algorithms reported in [Dietterich *et al.*, 1997] (iterated-discrim APR and GFS elim-kde APR), as well as the MULTINST algorithm from [Auer, 1997]. We note that the performances reported for iterated-discrim APR involves choosing parameters to maximize *test set* performance and so probably represents an upper bound for accuracy on this data set. The MULTINST algorithm assumes that all instances from all bags are generated independently. The Diverse Density results, which required no tuning, are comparable or better than those of GFS elim-kde APR and MULTINST.

| Musk Data Set 1 | | Musk Data Set 2 | |
| --- | --- | --- | --- |
| algorithm | accuracy | algorithm | accuracy |
| iterated-discrim APR | 92.4 | iterated-discrim APR | 89.2 |
| GFS elim-kde APR | 91.3 | MULTINST | 84.0 |
| Diverse Density | 88.9 | Diverse Density | 82.5 |
| MULTINST | 76.7 | GFS elim-kde APR | 80.4 |

We also investigated two new applications of multiple-instance learning. The first of these is to learn a simple description of a person from a series of images that are labeled positive if they contain the person and negative otherwise. For a positively labeled image we only know that the person is somewhere in it, but we do not know where. We sample 54 subimages of varying centers and sizes and declare them to be instances in one positive bag since one of them contains the person. This is repeated for every positive and negative image.

We use a very simple representation for the instances. Each subimage is divided into three parts which roughly correspond to where the head, torso and legs of the person would be. The three dominant colors (one for each subsection) are used to represent the image. Figure 4 shows a training set where every bag included two people, yet the algorithm learned a description of the person who appears in all the images. This technique is expanded in [Maron and LakshmiRatan, 1998] to learn descriptions of natural images and use the learned concept to retrieve similar images from a large image database.

Another new application uses Diverse Density in the stock selection problem. Every month, there are stocks that perform well for fundamental reasons and stocks that perform well because of flukes; there are many more of the latter, but we are interested in the former. For every month, we take the 100 stocks with the highest return and put them in a positive bag, hoping that at least one of them did well for fundamental reasons. Negative bags are created from the bottom 5 stocks in every month. A stock instance is described by 17 features such as momentum, price to fair-value, etc. Grantham, Mayo, Van Otterloo & Co. kindly provided us with data on the 600 largest US stocks since 1978. We tested the algorithm through five runs of training for ten years, then testing on the next year. In each run, the algorithm returned the stock description (location in feature space and a scaling of the features) that maximized Diverse Density. The test stocks were then ranked and decilized by distance (in weighted feature space) to the max-DD point. Figure 5 shows the average return of every decile. The return in the top decile (stocks that are most like the "fundamental stock") is positive and

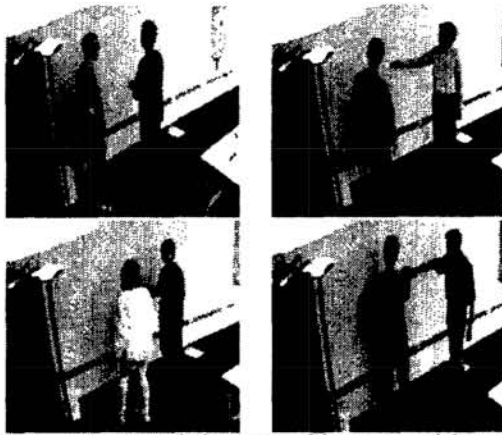

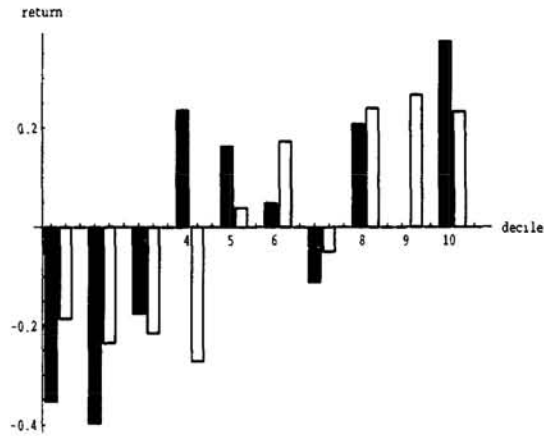

Figure 5: A training set of images with one person in common.

Figure 6: Black bars show Diverse Density's average return on a decile, and the white bars show GMO's predictor's return.

higher than the average return of a GMO predictor. Likewise, the return in the bottom decile is negative and below that of a GMO predictor.

## 4   Conclusion

In this paper, we have shown that Diverse Density is a general tool with which to learn from Multiple-Instance examples. In addition, we have shown that Multiple-Instance problems occur in a wide variety of domains. We attempted to show the various ways in which ambiguity can lead to the Multiple-Instance framework: through lack of knowledge in the drug discovery example, through ambiguity of representation in the vision example, and through a high degree of noise in the stock example.

### Acknowledgements
We thank Peter Dayan and Paul Viola at MIT and Tom Hancock and Chris Darnell at GMO for helpful discussions and the AFOSR ASSERT program, Parent Grant#:F49620-93-1-0263 for their support of this research.

## Footnotes

[1]In practice, one needs to restrict consideration to shapes of the molecule that have sufficiently low potential energy. But, we ignore this restriction in this simple illustration.

## References

[Auer, 1997] P. Auer. On Learning from Multi-Instance Examples: Empirical Evaluation of a theoretical Approach. NeuroCOLT Technical Report Series, NC-TR-97-025, March 1997.

[Blum and Kalai, 1998] A. Blum and A. Kalai. A Note on Learning from Multiple-Instance Examples. *To appear in Machine Learning*, 1998.

[Dietterich *et al.*, 1997] T. G. Dietterich, R. H. Lathrop, and T. Lozano-Pérez. Solving the Multiple-Instance Problem with Axis-Parallel Rectangles. *Artificial Intelligence Journal*, 89, 1997.

[Long and Tan, 1996] P. M. Long and L. Tan. PAC-learning axis alligned rectangles with respect to product distributions from multiple-instance examples. In *Proceedings of the 1996 Conference on Computational Learning Theory*, 1996.

[Maron and LakshmiRatan, 1998] O. Maron and A. LakshmiRatan. Multiple-Instance Learning for Natural Scene Classification. In *Submitted to CVPR-98*, 1998.